# Neural Net and Traditional Classifiers[1]

*William Y. Huang* and *Richard P. Lippmann*

MIT Lincoln Laboratory
Lexington, MA 02173, USA

**Abstract.** Previous work on nets with continuous-valued inputs led to generative procedures to construct convex decision regions with two-layer perceptrons (one hidden layer) and arbitrary decision regions with three-layer perceptrons (two hidden layers). Here we demonstrate that two-layer perceptron classifiers trained with back propagation can form both convex and disjoint decision regions. Such classifiers are robust, train rapidly, and provide good performance with simple decision regions. When complex decision regions are required, however, convergence time can be excessively long and performance is often no better than that of $k$-nearest neighbor classifiers. Three neural net classifiers are presented that provide more rapid training under such situations. Two use fixed weights in the first one or two layers and are similar to classifiers that estimate probability density functions using histograms. A third "feature map classifier" uses both unsupervised and supervised training. It provides good performance with little supervised training in situations such as speech recognition where much unlabeled training data is available. The architecture of this classifier can be used to implement a neural net $k$-nearest neighbor classifier.

## 1. INTRODUCTION

Neural net architectures can be used to construct many different types of classifiers [7]. In particular, multi-layer perceptron classifiers with continuous valued inputs trained with back propagation are robust, often train rapidly, and provide performance similar to that provided by Gaussian classifiers when decision regions are convex [12,7,5,8]. Generative procedures demonstrate that such classifiers can form convex decision regions with two-layer perceptrons (one hidden layer) and arbitrary decision regions with three-layer perceptrons (two hidden layers) [7,2,9]. More recent work has demonstrated that two-layer perceptrons can form non-convex and disjoint decision regions. Examples of hand crafted two-layer networks which generate such decision regions are presented in this paper along with Monte Carlo simulations where complex decision regions were generated using back propagation training. These and previous simulations [5,8] demonstrate that convergence time with back propagation can be excessive when complex decision regions are desired and performance is often no better than that obtained with $k$-nearest neighbor classifiers [4]. These results led us to explore other neural net classifiers that might provide faster convergence. Three classifiers called, "fixed weight," "hypercube," and "feature map" classifiers, were developed and evaluated. All classifiers were tested on illustrative problems with two continuous-valued inputs and two classes (A and B). A more restricted set of classifiers was tested with vowel formant data.

## 2. CAPABILITIES OF TWO LAYER PERCEPTRONS

Multi-layer perceptron classifiers with hard-limiting nonlinearities (node outputs of 0 or 1) and continuous-valued inputs can form complex decision regions. Simple constructive proofs demonstrate that a three-layer perceptron (two hidden layers) can

[1] This work was sponsored by the Defense Advanced Research Projects Agency and the Department of the Air Force. The views expressed are those of the authors and do not reflect the policy or position of the U. S. Government.

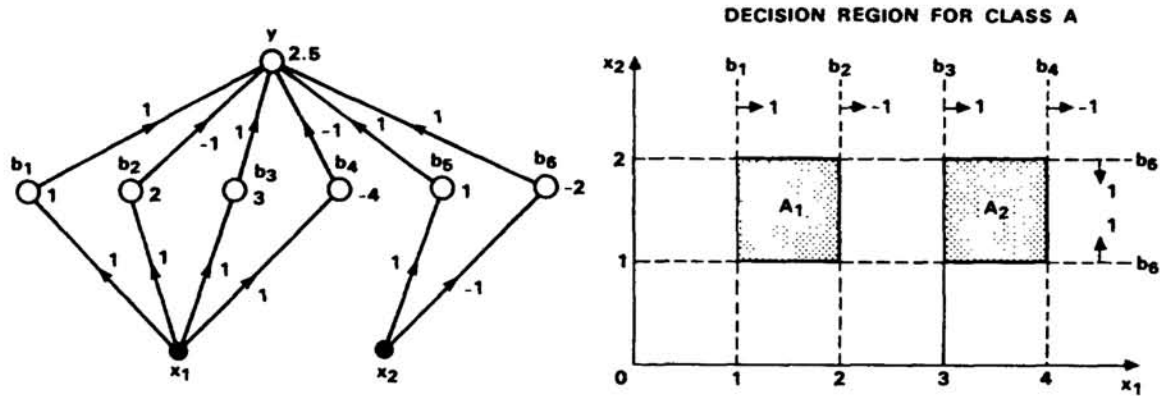

FIG. 1. *A two-layer perceptron that forms disjoint decision regions for class A (shaded areas). Connection weights and node offsets are shown in the left. Hyperplanes formed by all hidden nodes are drawn as dashed lines with node labels. Arrows on these lines point to the half plane where the hidden node output is "high".*

form arbitrary decision regions and a two-layer perceptron (one hidden layer) can form single convex decision regions [7,2,9]. Recently, however, it has been demonstrated that two-layer perceptrons can form decision regions that are not simply convex [14]. Fig. 1, for example, shows how disjoint decision regions can be generated using a two-layer perceptron. The two disjoint shaded areas in this Fig. represent the decision region for class A (output node has a "high" output, y = 1). The remaining area represents the decision region for class B (output node has a "low" output, y = 0). Nodes in this Fig. contain hard-limiting nonlinearities. Connection weights and node offsets are indicated in the left diagram. Ten other complex decision regions formed using two-layer perceptrons are presented in Fig. 2.

The above examples suggest that two-layer perceptrons can form decision regions with arbitrary shapes. We, however, know of no general proof of this capability. A 1965 book by Nilson discusses this issue and contains a proof that two-layer nets can divide a finite number of points into two arbitrary sets ([10] page 89). This proof involves separating $M$ points using at most $M - 1$ parallel hyperplanes formed by first-layer nodes where no hyperplane intersects two or more points. Proving that a given decision region can be formed in a two-layer net involves testing to determine whether the Boolean representations at the output of the first layer for all points within the decision region for class A are linearly separable from the Boolean representations for class B. One test for linear separability was presented in 1962 [13].

A problem with forming complex decision regions with two-layer perceptrons is that weights and offsets must be adjusted carefully because they interact extensively to form decision regions. Fig. 1 illustrates this sensitivity problem. Here it can be seen that weights to one hidden node form a hyperplane which influences decision regions in an entire half-plane. For example, small errors in first layer weights that results in a change in the slopes of hyperplanes $b_5$ and $b_6$ might only slightly extend the $A_1$ region but completely eliminate the $A_2$ region. This interdependence can be eliminated in three layer perceptrons.

It is possible to train two-layer perceptrons to form complex decision regions using back propagation and sigmoidal nonlinearities despite weight interactions. Fig. 3, for example, shows disjoint decision regions formed using back propagation for the problem of Fig. 1. In this and all other simulations, inputs were presented alternately from classes A and B and selected from a uniform distribution covering the desired decision region. In addition, the back propagation rate of descent term, $\eta$, was set equal to the momentum gain term, $\alpha$ and $\eta = \alpha = .01$. Small values for $\eta$ and $\alpha$ were necessary to guarantee convergence for the difficult problems in Fig. 2. Other simulation details are

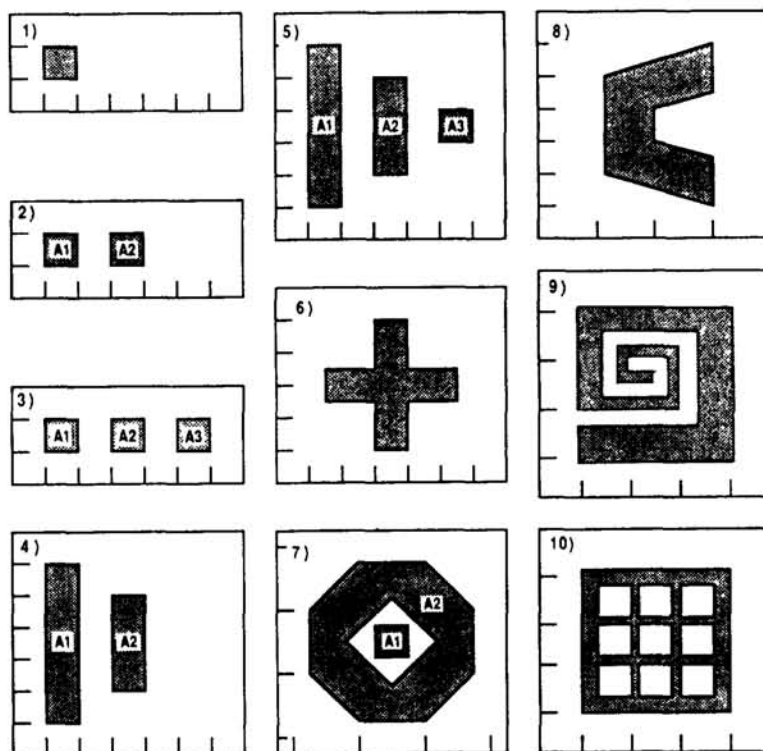

FIG. 2. *Ten complex decision regions formed by two-layer perceptrons. The numbers assigned to each case are the "case" numbers used in the rest of this paper.*

as in [5,8]. Also shown in Fig. 3 are hyperplanes formed by those first-layer nodes with the strongest connection weights to the output node. These hyperplanes and weights are similar to those in the networks created by hand except for sign inversions, the occurrence of multiple similar hyperplanes formed by two nodes, and the use of node offsets with values near zero.

## 3. COMPARATIVE RESULTS OF TWO-LAYERS VS. THREE-LAYERS

Previous results [5,8], as well as the weight interactions mentioned above, suggest that three-layer perceptrons may be able to form complex decision regions faster with back propagation than two-layer perceptrons. This was explored using Monte Carlo simulations for the first nine cases of Fig. 2. All networks have 32 nodes in the first hidden layer. The number of nodes in the second hidden layer was twice the number of convex regions needed to form the decision region (2, 4, 6, 4, 6, 6, 8, 6 and 6 for Cases 1 through 9 respectively). Ten runs were typically averaged together to obtain a smooth curve of percentage error vs. time (number of training trials) and enough trials were run (to a limit of 250,000) until the curve appeared to flatten out with little improvement over time. The error curve was then low-pass filtered to determine the convergence time. Convergence time was defined as the time when the curve crossed a value 5 percentage points above the final percentage error. This definition provides a framework for comparing the convergence time of the different classifiers. It, however, is not the time after which error rates do not improve. Fig. 4 summarizes results in terms of convergence time and final percentage error. In those cases with disjoint decision regions, back propagation sometimes failed to form separate regions after 250,000 trials. For example, the two disjoint regions required in Case 2 were never fully separated with

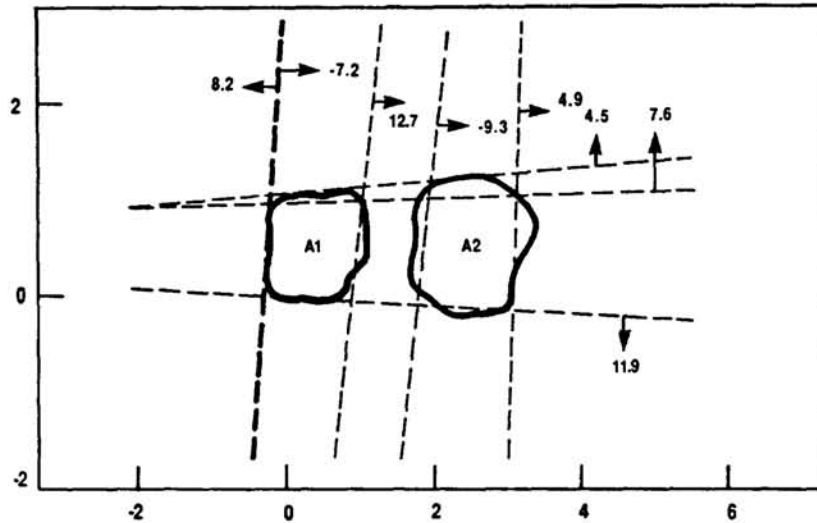

FIG. 3. *Decision regions formed using back propagation for Cases 2 of Fig. 2. Thick solid lines represent decision boundaries. Dashed lines and arrows have the same meaning as in Fig. 1. Only hyperplanes for hidden nodes with large weights to the output node are shown. Over 300,000 training trials were required to form separate regions.*

a two-layer perceptron but were separated with a three-layer perceptron. This is noted by the use of filled symbols in Fig. 4.

Fig. 4 shows that there is no significant performance difference between two and three layer perceptrons when forming complex decision regions using back propagation training. Both types of classifiers take an excessively long time ($> 100,000$ trials) to form complex decision regions. A minor difference is that in Cases 2 and 7 the two-layer network failed to separate disjoint regions after 250,000 trials whereas the three-layer network was able to do so. This, however, is not significant in terms of convergence time and error rate. Problems that are difficult for the two-layer networks are also difficult for the three-layer networks, and vice versa.

## 4. ALTERNATIVE CLASSIFIERS

Results presented above and previous results [5,8] demonstrate that multi-layer perceptron classifiers can take very long to converge for complex decision regions. Three alternative classifiers were studied to determine whether other types of neural net classifiers could provide faster convergence.

### 4.1. FIXED WEIGHT CLASSIFIERS

Fixed weight classifiers attempt to reduce training time by adapting only weights between upper layers of multi-layer perceptrons. Weights to the first layer are fixed before training and remain unchanged. These weights form fixed hyperplanes which can be used by upper layers to form decision regions. Performance will be good if the fixed hyperplanes are near the decision region boundaries that are required in a specific problem. Weights between upper layers are trained using back propagation as described above. Two methods were used to adjust weights to the first layer. Weights were adjusted to place hyperplanes randomly or in a grid in the region ($-1 < x_1, x_2 < 10$). All decision regions in Fig. 2 fall within this region. Hyperplanes formed by first layer nodes for "fixed random" and "fixed grid" classifiers for Case 2 of Fig. 2 are shown as dashed lines in Fig. 5. Also shown in this Fig. are decision regions (shaded areas) formed

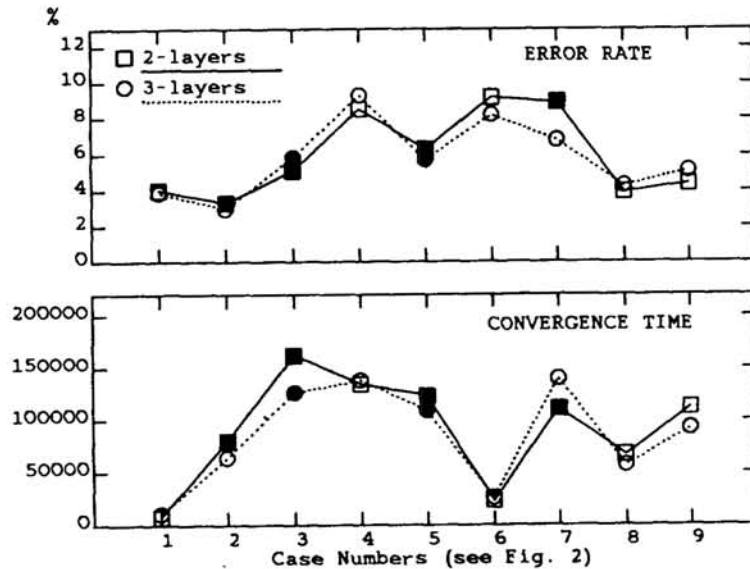

FIG. 4. *Percentage error (top) and convergence time (bottom) for Cases 1 through 9 of Fig. 2 for two-and three-layer perceptron classifiers trained using back propagation. Filled symbols indicate that separate disjoint regions were not formed after 250,000 trials.*

using back propagation to train only the upper network layers. These regions illustrate how fixed hyperplanes are combined to form decision regions. It can be seen that decision boundaries form along the available hyperplanes. A good solution is possible for the fixed grid classifier where desired decision region boundaries are near hyperplanes. The random grid classifier provides a poor solution because hyperplanes are not near desired decision boundaries. The performance of a fixed weight classifier depends both on the placement of hyperplanes and on the number of hyperplanes provided.

## 4.2. HYPERCUBE CLASSIFIER

Many traditional classifiers estimate probability density functions of input variables for different classes using histogram techniques [4]. Hypercube classifiers use this technique by fixing weights in the first two layers to break the input space into hypercubes (squares in the case of two inputs). Hypercube classifiers are similar to fixed weight classifiers, except weights to the first *two* layers are fixed, and only weights to output nodes are trained. Hypercube classifiers are also similar in structure to the CMAC model described by Albus [1]. The output of a second layer node is "high" only if the input is in the hypercube corresponding to that node. This is illustrated in Fig. 6 for a network with two inputs.

The top layer of a hypercube classifier can be trained using back propagation. A maximum likelihood approach, however, suggests a simpler training algorithm which consists of counting. The output of second layer node $H_i$ is connected to the output node corresponding to that class with greatest frequency of occurrence of training inputs in hypercube $H_i$. That is, if a sample falls in hypercube $H_i$, then it is classified as class $\theta^*$ where

$$N_{i,\theta^*} > N_{i,\theta} \quad \text{for all} \quad \theta \neq \theta^*. \tag{1}$$

In this equation, $N_{i,\theta}$ is the number of training tokens in hypercube $H_i$ which belong to class $\theta$. This will be called maximum likelihood (ML) training. It can be implemented by connection second-layer node $H_i$ only to that output node corresponding to class $\theta^*$ in Eq. (1). In all simulations hypercubes covered the area $(-1 < x_1, x_2 < 10)$.

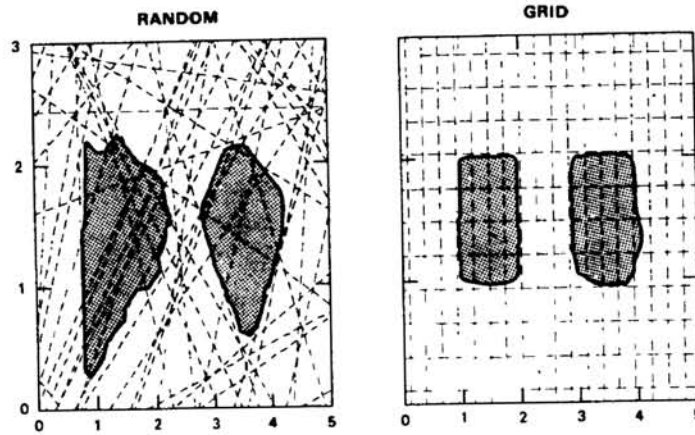

FIG. 5. *Decision regions formed with "fixed random" and "fixed grid" classifiers for Case 2 from Fig. 2 using back propagation training. Lines shown are hyperplanes formed by the first layer nodes. Shaded areas represent the decision region for class A.*

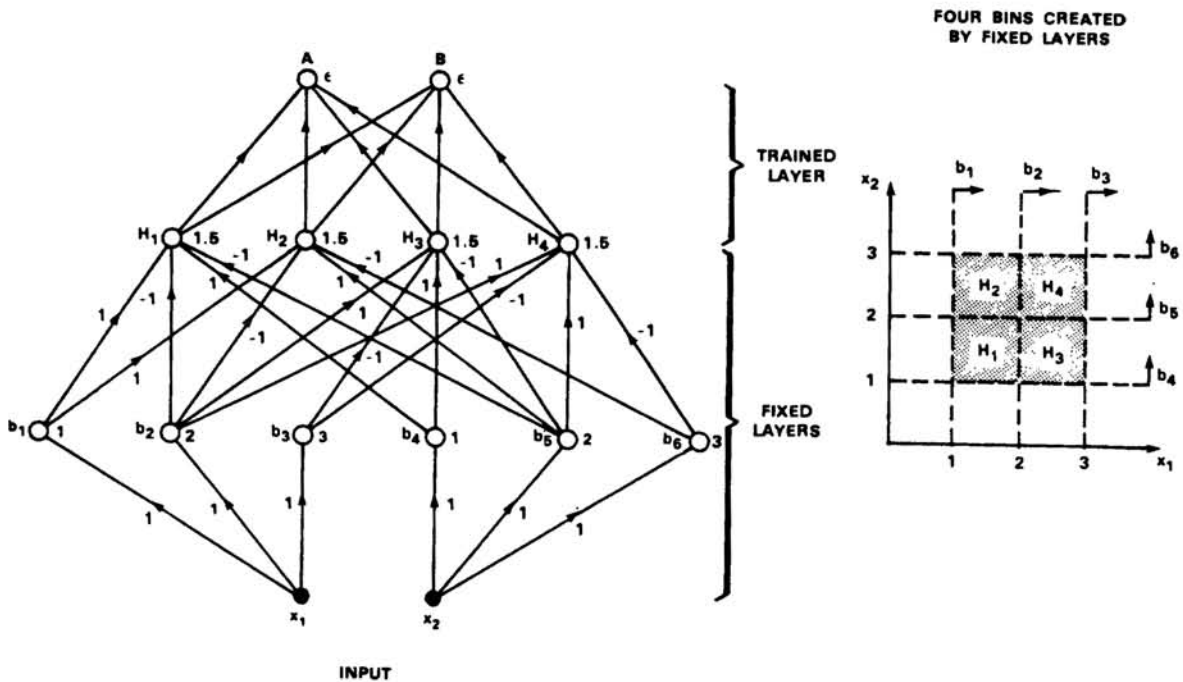

FIG. 6. *A hypercube classifier (left) is a three-layer perceptron with fixed weights to the first two layers, and trainable weights to output nodes. Weights are initialized such that outputs of nodes $H_1$ through $H_4$ (left) are "high" only when the input is in the corresponding hypercube (right).*

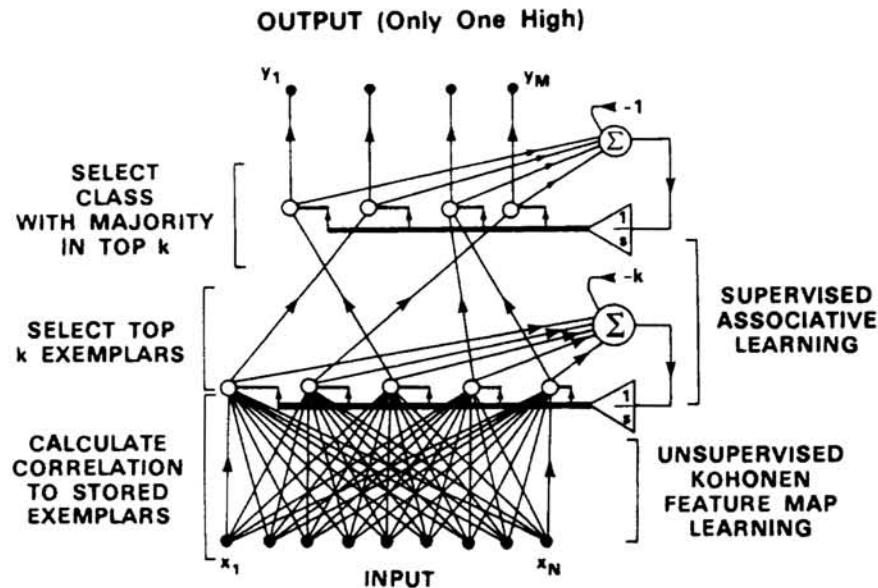

FIG. 7. *Feature map classifier.*

### 4.3. FEATURE MAP CLASSIFIER

In many speech and image classification problems a large quantity of unlabeled training data can be obtained, but little labeled data is available. In such situations unsupervised training with unlabeled training data can substantially reduce the amount of supervised training required [3]. The feature map classifier shown in Fig. 7 uses combined supervised/unsupervised training, and is designed for such problems. It is similar to histogram classifiers used in discrete observation hidden Markov models [11] and the classifier used in [6]. The first layer of this classifier forms a feature map using a self organizing clustering algorithm described by Kohonen [6]. In all simulations reported in this paper 10,000 trials of unsupervised training were used. After unsupervised training, first-layer feature nodes sample the input space with node density proportional to the combined probability density of all classes. First layer feature map nodes perform a function similar to that of second layer hypercube nodes except each node has maximum output for input regions that are more general than hypercubes and only the output of the node with a maximum output is fed to the output nodes. Weights to output nodes are trained with supervision after the first layer has been trained. Back propagation, or maximum likelihood training can be used. Maximum likelihood training requires $N_{i,\theta}$ (Eq. 1) to be the number of times first layer node $i$ has maximum output for inputs from class $\theta$. In addition, during classification, the outputs of nodes with $N_{i,\theta} = 0$ for all $\theta$ (untrained nodes) are not considered when the first-layer node with the maximum output is selected. The network architecture of a feature map classifier can be used to implement a $k$-nearest neighbor classifier. In this case, the feedback connections in Fig. 7 (large circular summing nodes and triangular integrators) used to select those $k$ nodes with the maximum outputs must be slightly modified. $K$ is 1 for a feature map classifier and must be adjusted to the desired value of $k$ for a $k$-nearest neighbor classifier.

## 5. COMPARISON BETWEEN CLASSIFIERS

The results of Monte Carlo simulations using all classifiers for Case 2 are shown in Fig. 8. Error rates and convergence times were determined as in Section 3. All alter-

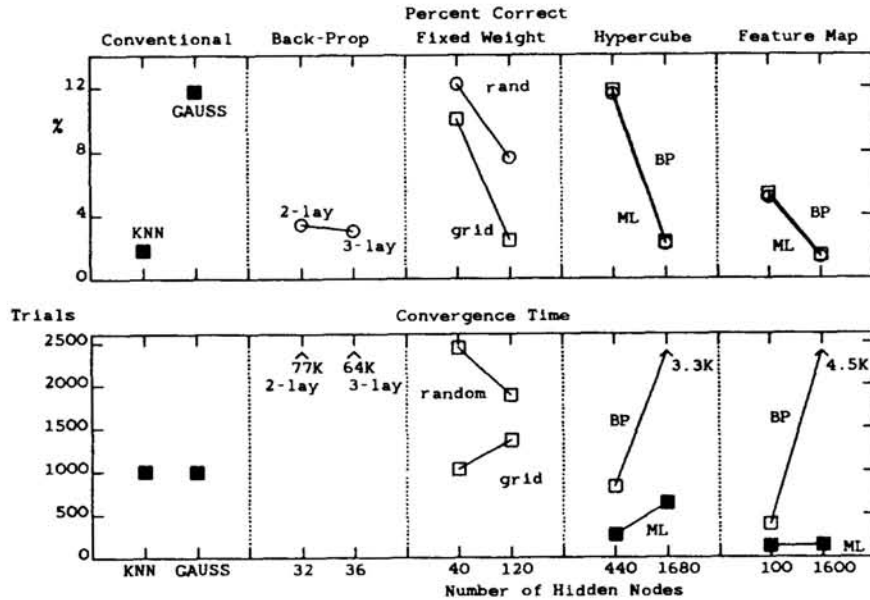

FIG. 8. *Comparative performance of classifiers for Case 2. Training time of the feature map classifiers does not include the 10,000 unsupervised training trials.*

native classifiers had shorter convergence times than multi-layer perceptron classifiers trained with back propagation. The feature map classifier provided best performance. With 1,600 nodes, its error rate was similar to that of the $k$-nearest neighbor classifiers but it required fewer than 100 supervised training tokens. The larger fixed weight and hypercube classifiers performed well but required more supervised training than the feature map classifiers. These classifiers will work well when the combined probability density function of all classes varies smoothly and the domain where this function is non-zero is known. In this case weights and offsets can be set such that hyperplanes and hypercubes cover the domain and provide good performance. The feature map classifier automatically covers the domain. Fixed weight "random" classifiers performed substantially worse than fixed weight "grid" classifiers. Back propagation training (BP) was generally much slower than maximum likelihood training (ML).

## 6. VOWEL CLASSIFICATION

Multi layer perceptron, feature map, and traditional classifiers were tested with vowel formant data from Peterson and Barney [11]. These data had been obtained by spectrographic analysis of vowels in /hVd/ context spoken by 67 men, women and children. First and second formant data of ten vowels was split into two sets, resulting in a total of 338 training tokens and 333 testing tokens. Fig. 9 shows the test data and the decision regions formed by a two-layer perceptron classifier trained with back propagation. The performance of classifiers is presented in Table I. All classifiers had similar error rates. The feature map classifier with only 100 nodes required less than 50 supervised training tokens (5 samples per vowel class) for convergence. The perceptron classifier trained with back propagation required more than 50,000 training tokens. The first stage of the feature map classifier and the multi-layer perceptron classifier were trained by randomly selecting entries from the 338 training tokens after labels had been removed and using tokens repetitively.

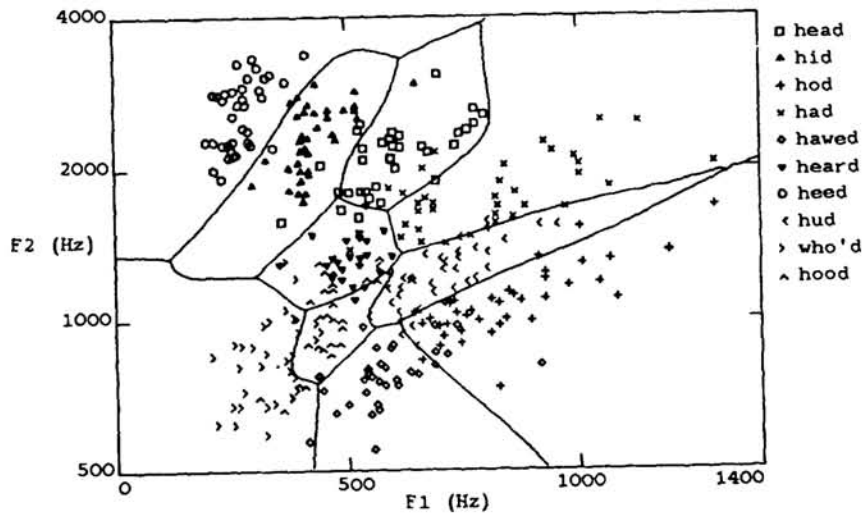

FIG. 9. *Decision regions formed by a two-layer network using BP after 200,000 training tokens from Peterson's steady state vowel data [Peterson, 1952]. Also shown are samples of the testing set. Legend show example of the pronunciation of the 10 vowels and the error within each vowel.*

| ALGORITHM | TRAINING TOKENS | % ERROR |
|---|---|---|
| KNN | 338 | 18.0 |
| Gaussian | 338 | 20.4 |
| 2-Layer Perceptron | 50,000 | 19.8 |
| Feature Map | < 50 | 22.8 |

TABLE I
*Performance of classifiers on steady state vowel data.*

## 7. CONCLUSIONS

Neural net architectures form a flexible framework that can be used to construct many different types of classifiers. These include Gaussian, $k$-nearest neighbor, and multi-layer perceptron classifiers as well as classifiers such as the feature map classifier which use unsupervised training. Here we first demonstrated that two-layer perceptrons (one hidden layer) can form non-convex and disjoint decision regions. Back propagation training, however, can be extremely slow when forming complex decision regions with multi-layer perceptrons. Alternative classifiers were thus developed and tested. All provided faster training and many provided improved performance. Two were similar to traditional classifiers. One (hypercube classifier) can be used to implement a histogram classifier, and another (feature map classifier) can be used to implement a $k$-nearest neighbor classifier. The feature map classifier provided best overall performance. It used combined supervised/unsupervised training and attained the same error rate as a $k$-nearest neighbor classifier, but with fewer supervised training tokens. Furthermore, it required fewer nodes then a $k$-nearest neighbor classifier.

## REFERENCES

[1] J. S. Albus, *Brains, Behavior, and Robotics.* McGraw-Hill, Petersborough, N.H., 1981.

[2] D. J. Burr, "A neural network digit recognizer," in *Proceedings of the International Conference on Systems, Man, and Cybernetics,* IEEE, 1986.

[3] D. B. Cooper and J. H. Freeman, "On the asymptotic improvement in the outcome of supervised learning provided by additional nonsupervised learning," *IEEE Transactions on Computers,* vol. C-19, pp. 1055–63, November 1970.

[4] R. O. Duda and P. E. Hart, *Pattern Classification and Scene Analysis.* John-Wiley & Sons, New York, 1973.

[5] W. Y. Huang and R. P. Lippmann, "Comparisons between conventional and neural net classifiers," in *1st International Conference on Neural Network,* IEEE, June 1987.

[6] T. Kohonen, K. Makisara, and T. Saramaki, "Phonotopic maps — insightful representation of phonological features for speech recognition," in *Proceedings of the 7th International Conference on Pattern Recognition,* IEEE, August 1984.

[7] R. P. Lippmann, "An introduction to computing with neural nets," *IEEE ASSP Magazine,* vol. 4, pp. 4–22, April 1987.

[8] R. P. Lippmann and B. Gold, "Neural classifiers useful for speech recognition," in *1st International Conference on Neural Network,* IEEE, June 1987.

[9] I. D. Longstaff and J. F. Cross, "A pattern recognition approach to understanding the multi-layer perceptron," Mem. 3936, Royal Signals and Radar Establishment, July 1986.

[10] N. J. Nilsson, *Learning Machines.* McGraw Hill, N.Y., 1965.

[11] T. Parsons, *Voice and Speech Processing.* McGraw-Hill, New York, 1986.

[12] F. Rosenblatt, *Perceptrons and the Theory of Brain Mechanisms.* Spartan Books, 1962.

[13] R. C. Singleton, "A test for linear separability as applied to self-organizing machines," in *Self-Organization Systems, 1962,* (M. C. Yovits, G. T. Jacobi, and G. D. Goldstein, eds.), pp. 503–524, Spartan Books, Washington, 1962.

[14] A. Wieland and R. Leighton, "Geometric analysis of neural network capabilities," in *1st International Conference on Neural Networks,* IEEE, June 1987.
